# Simple Local Models for Complex Dynamical Systems

**Erik Talvitie**
Computer Science and Engineering
University of Michigan
`etalviti@umich.edu`

**Satinder Singh**
Computer Science and Engineering
University of Michigan
`baveja@umich.edu`

## Abstract

We present a novel mathematical formalism for the idea of a "local model" of an uncontrolled dynamical system, a model that makes only certain predictions in only certain situations. As a result of its restricted responsibilities, a local model may be far simpler than a complete model of the system. We then show how one might combine several local models to produce a more detailed model. We demonstrate our ability to learn a collection of local models on a large-scale example and do a preliminary empirical comparison of learning a collection of local models and some other model learning methods.

## 1 Introduction

Building models that make good predictions about the world can be a complicated task. Humans, however, seem to have the remarkable ability to split this task up into manageable chunks. For instance, the activity in a park may have many complex interacting components (people, dogs, balls, etc.) and answering questions about their joint state would be impossible. It can be much simpler to answer abstract questions like "Where will the ball bounce?" ignoring most of the detail of what else might happen in the next moment. Some other questions like "What will the dog do?" may *still* be very difficult to answer in general, as dogs are complicated objects and their behavior depends on many factors. However, in certain situations, it may be relatively easy to make a prediction. If a ball has just been thrown, one may reasonably predict that the dog will chase it, without too much consideration of other potentially relevant facts. In short, it seems that humans have a lot of simple, *localized* pieces of knowledge that allow them to make predictions about particular aspects of the world in restricted situations. They can combine these abstract predictions to form more concrete, detailed predictions. Of course, there has been substantial effort in exploiting locality/independence structure in AI. Much of it is focused on static domains without temporal concerns (e.g. [1]), though these ideas have been applied in dynamical settings as well (e.g. [2, 3]). Our main contribution is to provide a novel mathematical formulation of "local models" of dynamical systems that make only certain predictions in only certain situations. We also show how to combine them into a more complete model. Finally, we present empirical illustrations of the use of our local models.

### 1.1 Background

In this paper we will focus on learning models of uncontrolled discrete dynamical systems (we leave consideration of controlled systems to future work). At each time step $i$ the system emits an observation $o_i$ from a finite set of observations $\mathcal{O}$. We call sequences of observations *tests* and let $\mathcal{T}$ be the set of all possible tests of all lengths. At time step $i$, the *history* is simply the sequence $o^1 o^2 ... o^i$ of past observations. We use the letter $\phi$ to represent the *null history* in which no observation has yet been emitted. A *prediction* of a test $t = o^{i+1}...o^{i+k}$ given a history $h = o^1...o^i$, which we denote $p(t|h)$, is the conditional probability that the sequence $t$ will occur, given that the sequence $h$ has already occurred: $p(t|h) \stackrel{\text{def}}{=} \Pr(o_{i+1} = o^{i+1}, ..., o_{i+k} = o^{i+k} | o_1 = o^1, ..., o_i = o^i)$. The set of all histories $\mathcal{H}$ is defined: $\mathcal{H} \stackrel{\text{def}}{=} \{t \in \mathcal{T} : p(t|\phi) > 0\} \cup \{\phi\}$. We use *models* to make predictions:

**Definition 1.** A *complete model* can generate predictions $p(t|h)$ for all $t \in \mathcal{T}$ and $h \in \mathcal{H}$.

A model that can make every such prediction can make any conditional prediction about the system [4]. For instance, one may want to make predictions about whether any one of a set of possible futures will occur (e.g. "Will the man throw a ball any time before he leaves the park?"). We can represent this type of prediction using a *union test* (also called a "collective outcome" by Jaeger [5]).

**Definition 2.** A *union test* $T \subseteq \mathcal{T}$ is a set of tests such that if $t \in T$ then no prefix of $t$ is in $T$. The prediction of a union test is a sum of predictions: $p(T|h) \stackrel{\text{def}}{=} \sum_{t \in T} p(t|h)$.

Models may be provided by an expert, or we can learn them from experience with the system (in the form of a data set of observation sequences emitted by the system). The complexity of representing and learning a model often depends on the complexity of the system being modeled. The measure of complexity that we will adopt is called the *linear dimension* [6] and is defined as the rank of the "system dynamics matrix" (the infinite matrix of predictions whose $ij^{\text{th}}$ entry is $p(t_j|h_i)$ for all $t_j \in \mathcal{T}$ and $h_i \in \mathcal{H}$). It is also closely related to the number of underlying states in a Hidden Markov Model. We will not define it more formally here but note that when we say one system is simpler than another, we mean that it has a smaller linear dimension.

We will now present the main contributions of our work, starting by precisely defining a local model, and then showing how they can be combined to create a more complete model.

## 2 Local Models

In contrast to a complete model, a local model has limited prediction responsibilities and hence makes only certain predictions in certain situations.

**Definition 3.** Given a set of *tests of interest* $\mathcal{T}^I$ and a set of *histories of interest* $\mathcal{H}^I$, a *local model* is any model that generates the *predictions of interest*: $p(t|h)$ for all $t \in \mathcal{T}^I$ and $h \in \mathcal{H}^I$.

We will assume, in general, that the tests of interest are union tests. In this paper, we will place a constraint on $\mathcal{H}^I \subseteq \mathcal{H}$ which we will call the "semi-Markov" property, due to its close relationship to the concept of the same name in the "options" literature [7]; this assumption will be relaxed in future work. In words, we require that, in order to determine if the current history is of interest, we need only look at what has happened since the *preceeding* history of interest. Put formally,

**Definition 4.** A set of histories of interest $\mathcal{H}^I$ is *semi-Markov* iff $h, h' \in \mathcal{H}^I \cup \{\phi\}$ and $ht \in \mathcal{H}^I$ for some $t \in \mathcal{T}$, implies that either $h't \in \mathcal{H}^I$ or $p(h't|\phi) = 0$.

As a simple example, consider the *1D Ball Bounce* system (see Figure 1). The agent observes a line of pixels, one of which (the location of the "ball") is black; the rest are white. The ball moves along the line, changing direction when it hits the edge. Each time step, with probability $0.5$, the ball sticks in place, and with probability $0.5$ it moves one square in its current direction.

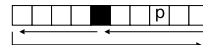

Figure 1: 1D Ball Bounce

One natural local model would make one-step predictions about only one pixel, $p$. It has two tests of interest: the set of all one-step tests in which the pixel $p$ is black, and the set of all one-step tests in which $p$ is white. All histories are of interest. This local model answers the question "What is the chance the ball will be in pixel $p$ next?" Note that, in order to answer this question, we need only observe the color of the pixels neighboring $p$. We will refer to this example as *Model A*.

Another, even more restricted local model would be one that has the same tests of interest, but whose histories of interest are only those that end with pixel $p$ being black. This local model would essentially answer the question "When the ball is in pixel $p$, what is the chance that it will stick?" In order to make this prediction, the local model can ignore *all* detail; the prediction for the test of interest is *always* $0.5$ at histories of interest. We will refer to this local model as *Model B*.

In general, as in the examples above, we expect that many details about the world are irrelevant to making the predictions of interest and could be ignored in order to simplify the local model. Taking an approach similar to that of, e.g., Wolfe & Barto [8], Soni & Singh [9], or Talvitie et al. [10], given tests and histories of interest, we will show how to convert a primitive observation sequence into an

*abstract* observation sequence that ignores unnecessary detail. A *complete* model of the abstracted system can be used as a *local* model in the original, primitive system. The abstraction proceeds in two steps (shown in Figure 2). First, we construct an intermediate system which makes predictions for all tests, but only updates at histories of interest. Then we further abstract the system by ignoring details irrelevant to making predictions for just the tests of interest.

## 2.1 Abstracting Details for Local Predictions

**Incorporating Histories Of Interest**: Intuitively, since a local model is never asked to make a prediction at a history outside of $\mathcal{H}^I$, one way to simplify it is to only update its predictions at histories of interest. Essentially, it "wakes up" whenever a history of interest occurs, sees what observation sequence happened since it was last awake, updates, and then goes dormant until the next history of interest. We call the sequences of observations that happen *between* histories of interest *bridging tests*. The set of bridging tests $\mathcal{T}^B$ is induced by the set of histories of interest.

**Definition 5.** A test $t \in \mathcal{T}$ is a *bridging test* iff for all $j < |t|$, and all $h \in \mathcal{H}^I$, $ht^{[1\ldots j]} \notin \mathcal{H}^I$ (where $t^{[1\ldots j]}$ denotes the $j$-length prefix of $t$) and either $\exists\, h \in \mathcal{H}^I$ such that $ht \in \mathcal{H}^I$ or $|t| = \infty$.

Conceptually, we transform the primitive observation sequence into a sequence of abstract observations in which each observation corresponds to a bridging test. We call such a transformed sequence the *Temporally Extended* or $TE$ sequence (see Figure 2). Note that even when the primitive system has a small number of observations, the $TE$ system can have infinitely many, because there can be an infinity of bridging tests. However, because it does not update between histories of interest, a model of $TE$ may be simpler than a model of the original system. To see this, consider again the 1D Ball Bounce of size $k$. This system has linear dimension $O(2k)$, in-

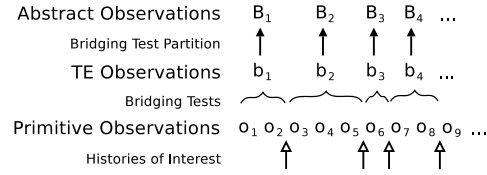

Figure 2: Mapping experience in the original system to experience in the TE system, and then to experience in the abstract system.

tuitively because the ball has 2 possible directions and $k$ possible positions. Recall *Model B*, that only applies when the ball lands on a particular pixel. The bridging tests, then, are all possible ways the ball could travel to an edge and back. The probability of each bridging test depends only on the current direction of the ball. As such, the $TE$ system here has linear dimension 2, regardless of $k$. It is possible to show formally that the $TE$ system is *never* more complex than the original system.

**Proposition 1.** *If the linear dimension of a dynamical system is $n$ then, given a semi-Markov set of histories of interest $\mathcal{H}^I$, the linear dimension of the induced $TE$ system, $n_{TE} \leq n$.*

*Proof.* (Sketch) The linear dimension of a system is the rank of the system dynamics matrix (SDM) corresponding to the system [6]. The matrix corresponding to the $TE$ system is the submatrix of the SDM of the original system with only columns and rows corresponding to histories and tests that are sequences of bridging tests. A submatrix never has greater rank than the matrix that contains it. □

What good is a model of the TE system? We next show that a model of the TE system can make predictions for all tests $t \in \mathcal{T}$ in all histories of interest $h \in \mathcal{H}^I$. Specifically, we show that the prediction for *any* test in a history of interest can be expressed as a prediction of a *union test* in $TE$. For the following, note that every history of interest $h \in \mathcal{H}^I$ can be written as a corresponding sequence of bridging tests, which we will call $s_h$. Also, we will use the subscript $TE$ to distinguish predictions $p_{TE}(t|h)$ in $TE$ from predictions $p(t|h)$ in the original system.

**Proposition 2.** *For any primitive test $t \in \mathcal{T}$ in the original system, there is a union test $S_t$ in $TE$ such that $p(t|h) = p_{TE}(S_t|s_h)$ for all $h \in \mathcal{H}^I$.*

*Proof.* We will present a constructive proof. First suppose $t$ can be written as a sequence of bridging tests $s_t$. Then trivially $S_t = \{s_t\}$. If $t$ does *not* correspond to a sequence of bridging tests, we can re-write it as the concatenation of two tests: $t = t_1 t_2$ such that $t_1$ is the longest prefix of $t$ that *is* a sequence of bridging tests (which may be null) and $t_2 \notin \mathcal{T}^B$. Now, $p(t|h) = p(t_1|h)p(t_2|ht_1)$, where $h, ht_1 \in \mathcal{H}^I$. We know already that $p(t_1|h) = p_{TE}(s_{t_1}|s_h)$. To calculate $p(t_2|ht_1)$ note that

there must be a *set* of bridging tests $B_{t_2}$ which have $t_2$ as a prefix: $B_{t_2} \stackrel{\text{def}}{=} \{b \in \mathcal{T}^B : b^{[1\ldots|t_2|]} = t_2\}$. The probability of seeing $t_2$ is the probability of seeing any of the bridging tests in $B_{t_2}$. Thus, at the history of interest $ht_1$, $p(t_2|ht_1) = \sum_{b \in B_{t_2}} p(b|ht_1) = \sum_{b \in B_{t_2}} p_{TE}(b|s_h s_{t_1})$. So, we let $S_t = \{s_{t_1} b : b \in B_{t_2}\}$, which gives us the result. $\hfill\square$

Since tests of interest are union tests, to make the prediction of interest $p(T|h)$ for some $T \in \mathcal{T}^I$ and $h \in \mathcal{H}^I$ using a model of $TE$, we have simply $p(T|h) = p_{TE}(S_T|s_h) = \sum_{t \in T} p_{TE}(S_t|s_h)$.

A model of $TE$ is simpler than a complete model of the system because it only makes predictions at histories of interest. However, it still makes predictions for *all* tests. We can further simplify our modeling task by focusing on predicting the tests of interest.

**Incorporating Tests of Interest:** Recall *Model A* from our example. Since all histories are of interest, bridging tests are single observations, and $TE$ is exactly equivalent to the original system. However, note that in order to make the predictions of interest, one must only know whether the ball is neighboring or on the pixel. So, we need only distinguish observations in which the ball is nearby, and we can group the rest into one abstract observation: "the ball is far from the pixel."

In general we will attempt to abstract away unnecessary details of bridging tests by aliasing bridging tests that are equivalent with respect to making the predictions of interest. Specifically, we will define a partition, or a many-to-one mapping, from $TE$ observations (the bridging tests $\mathcal{T}^B$) to *abstract* observations $\mathcal{A}$. We will then use a model of the abstract system with $\mathcal{A}$ as its observations (see Figure 2) as our local model. So, $\mathcal{A}$ must have the following properties: (1) we must be able to express the tests of interest as a union of sequences of abstract observations in $\mathcal{A}$ and (2) an abstracted history must contain enough detail to make accurate predictions for the tests of interest.

Let us first consider how to satisfy (1). For ease of exposition, we will discuss a special case. We assume that tests of interest are unions of one-step tests (i.e., for any $T \in \mathcal{T}^I$, $T \subseteq \mathcal{O}$) and that $\mathcal{T}^I$ partitions $\mathcal{O}$, so every observation is contained within exactly one test of interest. One natural example that satisfies this assumption is where the local model makes one-step predictions for a particular dimension of a vector-valued observation. There is no fundamental barrier to treating tests of interest that are arbitrary union tests, but the development of the general case is more complex.

Note that if a union test $T \subset \mathcal{O}$, then the equivalent $TE$ union test, $S_T$, consists of every bridging test that begins with an observation in $T$. So, if $\mathcal{T}^I$ partitions $\mathcal{O}$, then $\mathcal{S}^I \stackrel{\text{def}}{=} \{S_T : T \in \mathcal{T}^I\}$ partitions the bridging tests, $\mathcal{T}^B$, according to their first observation. As such, if we chose $\mathcal{A} = \mathcal{S}^I$, or any refinement thereof, we would satisfy criterion (1). However, $\mathcal{S}^I$ may not satisfy (2). For instance, in our 1D Ball Bounce, in order to make accurate predictions for one pixel it does not suffice to observe that pixel and ignore the rest. We must also distinguish the color of the neighboring pixels. This problem was treated explicitly by Talvitie et al. [10]. They define an *accurate partition*:

**Definition 6.** An observation abstraction $\mathcal{A}$ is *accurate with respect to* $\mathcal{T}^I$ iff for any two primitive histories $h^1 = o^1...o^k$ and $h^2 = o'^1...o'^k$ such that $\forall i\ o^i$ and $o'^i$ are contained within the same abstract observation $O^i \in \mathcal{A}$, we have $p(T|h^1) = p(T|h^2), \forall T \in \mathcal{T}^I$.

The system we are abstracting is $TE$, so the observations are bridging tests. We require an accurate refinement of $\mathcal{S}^I$. Any refinement of $\mathcal{S}^I$ satisfies criterion (1). Furthermore, an accurate refinement is one that only aliases two histories if they result in the same predictions for the tests of interest. Thus, we can use an abstract history to make *exactly* the same predictions for the tests of interest that we would make if we had access to the primitive history. So, an accurate refinement also satisfies criterion (2). Furthermore, an accurate refinement always exists, because the partition that distinguishes every observation is trivially accurate, though in general we expect to be able to abstract away *some* detail. Finally, a model of the abstract system may be far simpler than a model of the original system or the $TE$ system, and can be no more complex:

**Proposition 3.** *If the linear dimension of a dynamical system is $n$ then the linear dimension of any local model $\mathcal{M}$, $n_{\mathcal{M}} \leq n_{TE} \leq n$.*

*Proof.* (Sketch) The rows and columns of the SDM corresponding to an abstraction of $TE$ are linear combinations of rows and columns of the SDM of $TE$ [10]. So, the rank of the abstract SDM can be no more than the rank of the SDM for $TE$. $\hfill\square$

**Learning a local model:** We are given tests and histories of interest and an accurate abstraction. To learn a local model, we first translate the primitive trajectories into $TE$ trajectories using the histories of interest, and then translate the $TE$ trajectories into abstract trajectories using the accurate abstraction (as in Figure 2). We can then train *any* model on the abstracted data. In our experiments, we use POMDPs [11], PSRs [4], and low-order Markov models as local model representations.

## 2.2 Combining Local Models

Consider a collection of local models $\mathcal{M}$. Each local model $M \in \mathcal{M}$ has tests of interest $\mathcal{T}_M^I$, histories of interest $\mathcal{H}_M^I$, and is an exact model of the abstract system induced by a given accurate refinement, $\mathcal{A}_M$. At any history $h$, the set of models $\mathcal{M}_h \stackrel{\text{def}}{=} \{M \in \mathcal{M} : h \in \mathcal{H}_M^I\}$ is available to make predictions for their tests of interest. However, we may wish to make predictions that are not specifically of interest to any local model. In that case, we must combine the abstract, coarse predictions made by individual models into more fine-grained joint predictions. We will make a modeling assumption that allows us to efficiently combine the predictions of local models:

**Definition 7.** The local models in $\mathcal{M}_h$ are *mutually conditionally independent, given $h$* iff for any subset $\{M_1, M_2, ..., M_k\} \subseteq \mathcal{M}_h$, and any $T_1 \in \mathcal{T}_{M_1}^I, T_2 \in \mathcal{T}_{M_2}^I, ..., T_k \in \mathcal{T}_{M_k}^I$, the prediction of the intersection is equal to the product of the predictions: $p(\cap_{i=1}^k T_i | h) = \prod_{i=1}^k p(T_i | h)$.

A domain expert specifying the structure of a collection of local models should strive to satisfy this property as best as possible since, given this assumption, a collection of local models can be used to make many more predictions than can be made by each individual model. We can compute the predictions of finer-grained tests (intersections of tests of interest) by multiplying predictions together. We can also compute the predictions of unions of tests of interest using the standard formula: $\text{Pr}(A \cup B) = \text{Pr}(A) + \text{Pr}(B) - \text{Pr}(A \cap B)$. At any history $h$ for which $\mathcal{M}_h \neq \emptyset$, a collection of local models can be used to make predictions for any union test that can be constructed by unioning/intersecting the tests of interest of the models in $\mathcal{M}_h$. This may not include all tests. Of course making *all* predictions may not be practical, or necessary. A collection of local models can selectively focus on making the most important predictions well, ignoring or approximating less important predictions to save on representational complexity.

Of course, a collection of local models *can* be a complete model. For instance, note that any model that can make the predictions $p(o|h)$ for every $o \in \mathcal{O}$ and $h \in \mathcal{H}$ is a complete model. This is because every prediction can be expressed in terms of one-step predictions: $p(o^1...o^k|h) = p(o^1|h)p(o^2|ho^1)...p(o^k|ho^1...o^{k-1})$. As such, if every one-step test is expressible as an intersection of tests of interest of models in $\mathcal{M}_h$ at every $h$, then $\mathcal{M}$ is a complete model. That said, for a given $\mathcal{M}$, the mutual conditional independence property may or may not hold. If it does not, predictions made using $\mathcal{M}$ will be approximate, even if each local model in $\mathcal{M}$ makes its predictions of interest exactly. It would be useful, in future work, to explore bounds on the error of this approximation.

When learning a collection of local models in this paper, we assume that tests and histories of interest as well as an accurate refinement for each model are given. We then train each local model individually on abstract data. This is a fair amount of knowledge to assume as given, though it is analogous to providing the structure of a graphical model and learning only the distribution parameters, which is common practice. Automatically splitting a system into simple local models is an interesting, challenging problem, and ripe ground for future research. We hope that casting the structure learning problem in the light of our framework may illuminate new avenues to progress.

## 2.3 Relationship to Other Structured Representations

Here we briefly discuss a few especially relevant alternative modeling technologies that also aim to exploit local and independence structure in dynamical systems.

**DBNs:** The dynamic Bayes network (DBN) [2] is a representation that exploits conditional independence structure. The main difference between DBNs and our collection of local models is that DBNs specify independence structure over "hidden variables" whose values are never observed. Our representation expresses structure entirely in terms of predictions of observations. Thus our structural assumptions can be verified using statistical tests on the data while DBN assumptions cannot be directly verified. That said, a DBN *does* decompose its world state into a set of random variables. It

Table 1: Local model structure for the arcade game

| $\mathcal{H}^I_{\mathcal{M}}$: $\mathcal{M}$ applies when history ends with: | $\mathcal{T}^I_{\mathcal{M}}$: $\mathcal{M}$ makes one-step predictions for: | $\mathcal{A}_{\mathcal{M}}$: $\mathcal{M}$ additionally distinguishes bridging tests by: |
| --- | --- | --- |
| Ball hitting brick $b$ | Color of 6×4 pixels within $b$ | Type of special bricks hit and type of special brick most recently hit |
| Ball not hitting brick $b$ | Color of 6×4 pixels within $b$ | None |
| Ball in position $p$, coming from direction $d$ | Absence or presence of ball color in $6 \times 6$ pixels around $p$ | Configuration of bricks adjacent to $p$ in last step of bridging test |
| No brick in pixel $p$ and no ball near pixel $p$ | Color of pixel $p$ | None |

stores the conditional probability distribution for each variable, given the values in the previous time step. These distributions are like local models that make one-step predictions about their variable. For each variable, a DBN also specifies which other variables can be ignored when predicting its next value. This is essentially our accurate refinement, which identifies details a local model can ignore. Histories of interest are related to the concept of context-specific independence [12].

**Relational Models:** Relational models (e.g. [3]) treat the state of the world as a conjunction of predicates. The state evolves using "update rules," consisting of pre-conditions specifying when the rule applies and post-conditions (changes to the state). Update rules are essentially local models with pre and post-conditions playing the roles of histories and tests of interest. Relational models typically focus on Markov worlds. We address partial observability by essentially generalizing the "update rule." The main strength of relational models is that they include first-order variables in update rules, allowing for sophisticated parameter tying and generalization. We use parameter tying in our experiments, but do not incorporate the formalism of variables into our framework.

**Others:** Wolfe and Singh recently introduced the Factored PSR [13] which is essentially a special collection of local models. Also related are maximum entropy models (e.g. [14], [15]) which represent predictions as weighted products of features of the future and the past.

## 3 Experimental Results

**Large Scale Example:** In this section we present preliminary empirical results illustrating the application of collections of local models. Our first example is a modified, uncontrolled version of an arcade game (see Figure 3). The observations are $64 \times 42$ pixel images. In the image is a $2 \times 2$ pixel ball and a wall of $6 \times 4$ pixel bricks. After the ball hits a brick, the brick disappears. When the ball hits the bottom wall, it bounces at a randomly selected angle. An episode ends when there are no more bricks. In our version there are two types of "special bricks." After the ball hits a dark brick, all bricks require two hits rather than

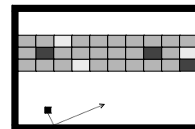

Figure 3: Arcade game

one to break. After the ball hits a light brick, all bricks require only one hit to break. When they are first placed, bricks are regular (medium gray) with probability 0.9 and dark or light each with probability 0.05. This system is stochastic, partially observable (and because of the special bricks, not short-order Markov). It has roughly $10^{20}$ observations and even more underlying states.

The decomposition into local models is specified in Table 1[1]. Quite naturally, we have local models to predict how the bricks (rows 1-2), the ball (row 3), and the background (row 4) will behave. This structure satisfies the mutual conditional independence property, and since every pixel is predicted by some model at every history, we can make fully detailed $64 \times 42$ pixel one-step predictions. More or less subdivision of models could be applied, the tradeoff being the complexity of individual models versus the total number of local models. With the structure we have selected there are approximately 25,000 local models. Of course, naively training 25,000 models is impractical. We can improve our data efficiency and training time though parameter tying. In this system, the behavior of objects does not depend on their position. To take advantage of this, for each *type* of local model

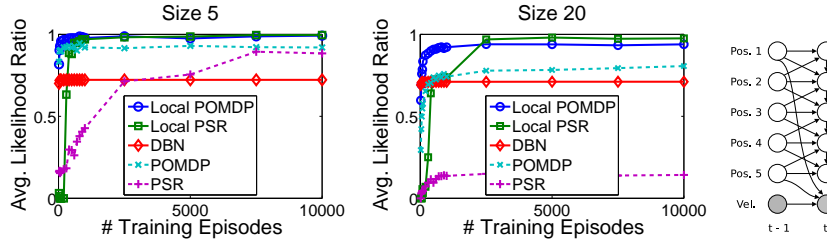

Figure 5: Left: Results for the 1D Ball Bounce problem. Error bars are omitted to avoid graph clutter. Right: DBN structure used. All nodes are binary. The shaded nodes are hidden. Links from "Vel." at $t-1$ to all nodes at $t$ omitted for simplicity.

(12 in total, since there is a ball model for each of the 9 directions) we combine all translated trajectories associated with various positions and use them to train a single shared model. Each local model maintains its own state, but the underlying model parameters are shared across all models of the same type, associated with different positions. Note that position does matter in the first time step, since the ball always appears in the same place. As a result, our model makes bad predictions about the first time step. For clarity of presentation, we will ignore the first time-step in our results.

For the local models themselves, we used lookup table based short-order Markov representations. Though the overall system is not short-order Markov, each local model is. Our learned local models were first-order Markov except the one responsible for predicting what will happen to a brick when the ball hits it. This model was second-order Markov. No local model had more than 200 states.

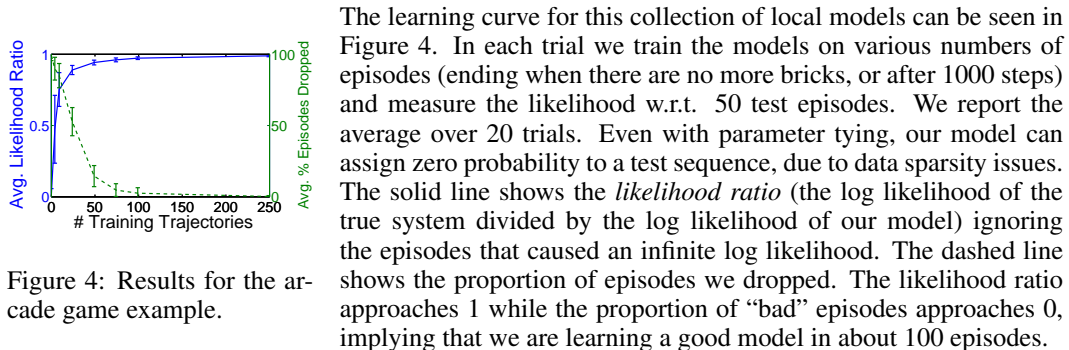

Figure 4: Results for the arcade game example.

The learning curve for this collection of local models can be seen in Figure 4. In each trial we train the models on various numbers of episodes (ending when there are no more bricks, or after 1000 steps) and measure the likelihood w.r.t. 50 test episodes. We report the average over 20 trials. Even with parameter tying, our model can assign zero probability to a test sequence, due to data sparsity issues. The solid line shows the *likelihood ratio* (the log likelihood of the true system divided by the log likelihood of our model) ignoring the episodes that caused an infinite log likelihood. The dashed line shows the proportion of episodes we dropped. The likelihood ratio approaches 1 while the proportion of "bad" episodes approaches 0, implying that we are learning a good model in about 100 episodes.

**Learning Comparisons:** In this experiment, we will compare parameter learning results for collections of local models to a few other methods on a simple example, whose complexity is easily controlled. Recall the 1D Ball Bounce. We learned a model of the 1D Ball Bounce of size 5 and 20 using two collections of local models with no parameter tying (using PSRs and POMDPs as local models respectively), two flat models (a PSR and a POMDP), and a DBN [2].

Both collections of local models have the following structure: for every pixel, there are two types of model. One predicts the color of the pixel in the next time step in histories when the ball is *not* in the immediate neighborhood about the pixel. This model ignores all pixels other than the one it is predicting. The other model applies when the ball *is* in the pixel. It jointly predicts the colors of the pixel and its two neighbors. This model distinguishes bridging tests in which the ball went to the left, the right, or stayed on the pixel in the first step. This collection of local models satisfies the mutual conditional independence property and allows prediction of primitive one-step tests.

As with the arcade game example, in each trial we trained each model on various numbers of episodes (of length 50) and then measured their log likelihood on 1000 test episodes (also of length

50). We report the likelihood ratio averaged over 20 trials. The results are shown in Figure 5. The collections of local models both perform well, outperforming the flat models (dashed lines). Both of the flat models' performance degrades as the size of the world increases from 5 to 20. The collections of local models are less affected by problem size. The local PSRs seem to take more data than the local POMDPs to learn a good model, however they ultimately seem to learn a better model. The unexpected result is that DBN training seemed to perform worse than flat POMDP training. We have no explanation for this effect, other than the fact that different graphical structures could cause different local extrema issues for the EM algorithm. Clearly, given these results, a more thorough empirical comparison across a wider variety of problems is warranted.

**Conclusions:** We have presented a novel formalization of the idea of a "local model." Preliminary empirical results show that collections of local models can be learned for large-scale systems and that the data complexity of parameter learning compares favorably to that of other representations.

### Acknowledgments

Erik Talvitie was supported under the NSF GRFP. Satinder Singh was supported by NSF grant IIS-0413004. Any opinions, findings, and conclusions or recommendations expressed in this material are those of the authors and do not necessarily reflect the views of the NSF.

## Footnotes

[1]Note: there are 30 bricks $b$, 2,688 pixels $p$, 2,183 possible positions $p$ for the ball, and 9 possible directions $d$ the ball could come from, including the case in the first step, where the ball simply appears in a pixel.

[2]We initialized each local POMDP with 5 states and the flat POMDP with 10 and 40 states for the different problem sizes. For the DBN we used the graphical structure shown in Figure 5(c) and trained using the Graphical Models Toolkit [16]. We stopped EM after a maximum of 50 iterations. PSR training also has a free parameter (see [17] for details). Via parameter sweep we chose 0.02 for local PSRs and for the flat PSR 0.175 and 0.005, respectively for the size 5 and size 20 domains.

## References

[1] Lise Getoor, Nir Friedman, Daphne Koller, and Benjamin Taskar. Learning probabilistic models of relational structure. *Journal of Machine Learning Research*, 3:679–707, 2002.

[2] Zoubin Ghahramani and Michael I. Jordan. Factorial hidden Markov models. In *Advances in Neural Information Processing Systems 8 (NIPS)*, pages 472–478, 1995.

[3] Hanna M. Pasula, Luke S. Zettlemoyer, and Leslie Pack Kaelbling. Learning symbolic models of stochastic domains. *Journal of Artificial Intelligence*, 29:309–352, 2007.

[4] Michael Littman, Richard Sutton, and Satinder Singh. Predictive representations of state. In *Advances in Neural Information Processing Systems 14 (NIPS)*, pages 1555–1561, 2002.

[5] Herbert Jaeger. Observable operator models for discrete stochastic time series. *Neural Computation*, 12(6):1371–1398, 2000.

[6] Satinder Singh, Michael R. James, and Matthew R. Rudary. Predictive state representations: A new theory for modeling dynamical systems. In *Uncertainty in Artificial Intelligence 20 (UAI)*, pages 512–519, 2004.

[7] Richard Sutton, Doina Precup, and Satinder Singh. Between mdps and semi-mdps: A framework for temporal abstraction in reinforcement learning. *Artificial Intelligence*, 112:181–211, 1999.

[8] Alicia Peregrin Wolfe and Andrew G. Barto. Decision tree methods for finding reusable MDP homomorphisms. In *National Conference on Artificial Intelligence 21 (AAAI)*, 2006.

[9] Vishal Soni and Satinder Singh. Abstraction in predictive state representations. In *National Conference on Artificial Intelligence 22 (AAAI)*, 2007.

[10] Erik Talvitie, Britton Wolfe, and Satinder Singh. Building incomplete but accurate models. In *International Symposium on Artificial Intelligence and Mathematics (ISAIM)*, 2008.

[11] George E. Monahan. A survey of partially observable markov decisions processes: Theory, models, and algorithms. *Management Science*, 28(1):1–16, 1982.

[12] Craig Boutilier, Nir Friedman, Moises Goldszmidt, and Daphne Koller. Context-specific independence in bayesian networks. In *Uncertainty in Artificial Intelligence 12 (UAI)*, pages 115–123, 1996.

[13] Britton Wolfe, Michael James, and Satinder Singh. Approximate predictive state representations. In *Autonomous Agents and Multiagent Systems 7 (AAMAS)*, 2008.

[14] Adam Berger, Stephen Della Pietra, and Vincent Della Pietra. A maximum entropy approach to natural language processing. *Computational Linguistics*, 22(1):39–71, 1996.

[15] David Wingate and Satinder Singh. Exponential family predictive representations of state. In *Advances in Neural Information Processing Systems 20 (NIPS)*, pages 1617–1624, 2007.

[16] Jeff Bilmes. The graphical models toolkit (gmtk), 2007. `http://ssli.ee.washington.edu/~bilmes/gmtk`.

[17] Michael James and Satinder Singh. Learning and discovery of predictive state representations in dynamical systems with reset. In *International Conference on Machine Learning 21 (ICML)*, 2004.

